# An Actor/Critic Algorithm that is Equivalent to Q-Learning

**Robert H. Crites**
Computer Science Department
University of Massachusetts
Amherst, MA 01003
crites@cs.umass.edu

**Andrew G. Barto**
Computer Science Department
University of Massachusetts
Amherst, MA 01003
barto@cs.umass.edu

## Abstract

We prove the convergence of an actor/critic algorithm that is equivalent to Q-learning by construction. Its equivalence is achieved by encoding Q-values within the policy and value function of the actor and critic. The resultant actor/critic algorithm is novel in two ways: it updates the critic only when the most probable action is executed from any given state, and it rewards the actor using criteria that depend on the relative probability of the action that was executed.

## 1 INTRODUCTION

In actor/critic learning systems, the actor implements a stochastic policy that maps states to action probability vectors, and the critic attempts to estimate the value of each state in order to provide more useful reinforcement feedback to the actor. The result is two interacting adaptive processes: the actor adapts to the critic, while the critic adapts to the actor.

The foundations of actor/critic learning systems date back at least to Samuel's checker program in the late 1950s (Samuel,1963). Examples of actor/critic systems include Barto, Sutton, & Anderson's (1983) ASE/ACE architecture and Sutton's (1990) Dyna-PI architecture. Sutton (1988) notes that the *critic* in these systems performs temporal credit assignment using what he calls *temporal difference* (TD) methods. Barto, Sutton, & Watkins (1990) note a relationship between actor/critic

architectures and a dynamic programming (DP) algorithm known as *policy iteration.*

Although DP is a collection of general methods for solving Markov decision processes (MDPs), these algorithms are computationally infeasible for problems with very large state sets. Indeed, classical DP algorithms require multiple complete sweeps of the entire state set. However, progress has been made recently in developing asynchronous, incremental versions of DP that can be run online concurrently with control (Watkins, 1989; Barto *et al*, 1993). Most of the theoretical results for incremental DP have been for algorithms based on a DP algorithm known as *value iteration.* Examples include Watkins' (1989) Q-learning algorithm (motivated by a desire for on-line learning), and Bertsekas & Tsitsiklis' (1989) results on asynchronous DP (motivated by a desire for parallel implementations). Convergence proofs for incremental algorithms based on *policy iteration* (such as actor/critic algorithms) have been slower in coming.

Williams & Baird (1993) provide a valuable analysis of the convergence of certain actor/critic learning systems that use deterministic policies. They assume that a model of the MDP (including all the transition probabilities and expected rewards) is available, allowing the use of operations that look ahead to all possible next states. When a model is not available for the evaluation of alternative actions, one must resort to other methods for exploration, such as the use of stochastic policies. We prove convergence for an actor/critic algorithm that uses stochastic policies and does not require a model of the MDP.

The key idea behind our proof is to construct an actor/critic algorithm that is equivalent to Q-learning. It achieves this equivalence by encoding Q-values within the policy and value function of the actor and critic. By illustrating the way Q-learning appears as an actor/critic algorithm, the construction sheds light on two significant differences between Q-learning and traditional actor/critic algorithms. Traditionally, the critic attempts to provide feedback to the actor by estimating $V^\pi$, the value function corresponding to the current policy $\pi$. In our construction, instead of estimating $V^\pi$, the critic directly estimates the optimal value function $V^*$. In practice, this means that the value function estimate $\hat{V}$ is updated only when the *most probable* action is executed from any given state. In addition, our actor is provided with more discriminating feedback, based not only on the TD error, but also on the relative probability of the action that was executed. By adding these modifications, we can show that this algorithm behaves exactly like Q-learning constrained by a particular exploration strategy. Since a number of proofs of the convergence of Q-learning already exist (Tsitsiklis, 1994; Jaakkola *et al*, 1993; Watkins & Dayan, 1992), the fact that this algorithm behaves exactly like Q-learning implies that it too converges to the optimal value function with probability one.

## 2   MARKOV DECISION PROCESSES

Actor/critic and Q-learning algorithms are usually studied within the Markov decision process framework. In a finite MDP, at each discrete time step, an agent observes the state $x$ from a finite set $X$, and selects an action $a$ from a finite set $A_x$ by using a stochastic policy $\pi$ that assigns a probability to each action in $A_x$. The agent receives a reward with expected value $R(x, a)$, and the state at the next

time step is $y$ with probability $p^a(x,y)$. For any policy $\pi$ and $x \in X$, let $V^\pi(x)$ denote the *expected infinite-horizon discounted return* from $x$ given that the agent uses policy $\pi$. Letting $r_t$ denote the reward at time $t$, this is defined as:

$$V^\pi(x) = E_\pi \left[ \sum_{t=0}^\infty \gamma^t r_t | x_0 = x \right], \tag{1}$$

where $x_0$ is the initial state, $0 \leq \gamma < 1$ is a factor used to discount future rewards, and $E_\pi$ is the expectation assuming the agent always uses policy $\pi$. It is usual to call $V^\pi(x)$ the *value* of $x$ under $\pi$. The function $V^\pi$ is the *value function* corresponding to $\pi$. The objective is to find an optimal policy, i.e., a policy, $\pi^*$, that maximizes the value of each state $x$ defined by (1). The unique *optimal value function*, $V^*$, is the value function corresponding to any optimal policy. Additional details on this and other types of MDPs can be found in many references.

## 3   ACTOR/CRITIC ALGORITHMS

A generic actor/critic algorithm is as follows:

1. Initialize the stochastic policy and the value function estimate.

2. From the current state $x$, execute action $a$ randomly according to the current policy. Note the next state $y$, the reward $r$, and the TD error

   $$\varepsilon = [r + \gamma \hat{V}(y)] - \hat{V}(x),$$

   where $0 \leq \gamma < 1$ is the discount factor.

3. Update the actor by adjusting the action probabilities for state $x$ using the TD error. If $\varepsilon > 0$, action $a$ performed relatively well and its probability should be increased. If $\varepsilon < 0$, action $a$ performed relatively poorly and its probability should be decreased.

4. Update the critic by adjusting the estimated value of state $x$ using the TD error:

   $$\hat{V}(x) \leftarrow \hat{V}(x) + \alpha\, \varepsilon$$

   where $\alpha$ is the learning rate.

5. $x \leftarrow y$. Go to step 2.

There are a variety of implementations of this generic algorithm in the literature. They differ in the exact details of how the policy is stored and updated. Barto *et al* (1990) and Lin (1993) store the action probabilities indirectly using parameters $w(x,a)$ that need not be positive, and need not sum to one. Increasing (or decreasing) the probability of action $a$ in state $x$ is accomplished by increasing (or decreasing) the value of the parameter $w(x,a)$. Sutton (1990) modifies the generic algorithm so that these parameters can be interpreted as action value estimates. He redefines $\varepsilon$ in step 2 as follows:

$$\varepsilon = [r + \gamma \hat{V}(y)] - w(x,a).$$

For this reason, the Dyna-PI architecture (Sutton, 1990) and the modified actor/critic algorithm we present below both reward less probable actions more readily because of their lower estimated values.

Barto *et al* (1990) select actions by adding exponentially distributed random numbers to each parameter $w(x, a)$ for the current state, and then executing the action with the maximum sum. Sutton (1990) and Lin (1993) convert the parameters $w(x, a)$ into action probabilities using the Boltzmann distribution, where given a temperature $T$, the probability of selecting action $i$ in state $x$ is

$$\frac{e^{w(x,i)/T}}{\sum_{a \in A_x} e^{w(x,a)/T}}.$$

In spite of the empirical success of these algorithms, their convergence has never been proven.

## 4   Q-LEARNING

Rather than learning the values of states, the Q-learning algorithm learns the values of state/action pairs. $Q(x, a)$ is the expected discounted return obtained by performing action $a$ in state $x$ and performing optimally thereafter. Once the $Q$ function has been learned, an optimal action in state $x$ is any action that maximizes $Q(x, \cdot)$. Whenever an action $a$ is executed from state $x$, the Q-value estimate for that state/action pair is updated as follows:

$$\hat{Q}(x, a) \leftarrow \hat{Q}(x, a) + \alpha_{xa}(n) \left[ r + \gamma \max_{b \in A_y} \hat{Q}(y, b) - \hat{Q}(x, a) \right],$$

where $\alpha_{xa}(n)$ is the non-negative learning rate used the $n$th time action $a$ is executed from state $x$. Q-Learning does not specify an exploration mechanism, but requires that all actions be tried infinitely often from all states. In actor/critic learning systems, exploration is fully determined by the action probabilities of the actor.

## 5   A MODIFIED ACTOR/CRITIC ALGORITHM

For each value $v \in \Re$, the modified actor/critic algorithm presented below uses an invertible function, $H_v$, that assigns a real number to each action probability ratio:

$$H_v : (0, \infty) \rightarrow \Re.$$

Each $H_v$ must be a continuous, strictly increasing function such that $H_v(1) = v$, and

$$H_{H_v(z_2)}\left(\frac{z_1}{z_2}\right) = H_v(z_1) \text{ for all } z_1, z_2 > 0.$$

One example of such a class of functions is $H_v(z) = T \, ln(z) + v$, $v \in \Re$, for some positive $T$. This class of functions corresponds to Boltzmann exploration in Q-learning. Thus, a kind of simulated annealing can be accomplished in the modified actor/critic algorithm (as is often done in Q-learning) by gradually lowering the "temperature" $T$ and appropriately renormalizing the action probabilities. It is also possible to restrict the range of $H_v$ if bounds on the possible values for a given MDP are known *a priori*.

For a state $x$, let $p_a$ be the probability of action $a$, let $p_{max}$ be the probability of the most probable action, $a_{max}$, and let $z_a = \frac{p_a}{p_{max}}$.

The modified actor/critic algorithm is as follows:

1. Initialize the stochastic policy and the value function estimate.

2. From the current state $x$, execute an action randomly according to the current policy. Call it action $i$. Note the next state $y$ and the immediate reward $r$, and let

$$\varepsilon = [r + \gamma \hat{V}(y)] - H_{\hat{V}(x)}(z_i).$$

3. Increase the probability of action $i$ if $\varepsilon > 0$, and decrease its probability if $\varepsilon < 0$. The precise probability update is as follows. First calculate

$$z_i^\star = H_{\hat{V}(x)}^{-1}[H_{\hat{V}(x)}(z_i) + \alpha_{xi}(n)\,\varepsilon].$$

Then determine the new action probabilities by dividing by normalization factor $N = z_i^\star + \sum_{j \neq i} z_j$, as follows:

$$p_i \leftarrow \frac{z_i^\star}{N}, \quad and \quad p_j \leftarrow \frac{z_j}{N}, \quad j \neq i.$$

4. Update $\hat{V}(x)$ only if $i = a_{max}$. Since the action probabilities are updated after every action, the most probable action may be different before and after the update. If $i = a_{max}$ both before *and* after step 3 above, then update the value function estimate as follows:

$$\hat{V}(x) \leftarrow \hat{V}(x) + \alpha_{xi}(n)\,\varepsilon$$

Otherwise, if $i = a_{max}$ before *or* after step 3:

$$\hat{V}(x) \leftarrow H_{\hat{V}(x)}(Np_k),$$

where action $k$ is the most probable action after step 3.

5. $x \leftarrow y$. Go to step 2.

## 6   CONVERGENCE OF THE MODIFIED ALGORITHM

**Theorem:**   *The modified actor/critic algorithm given above converges to the optimal value function $V^*$ with probability one if:*

1. *The state and action sets are finite.*

2. *$\sum_{n=0}^{\infty} \alpha_{xa}(n) = \infty$ and $\sum_{n=0}^{\infty} \alpha_{xa}^2(n) < \infty$.*

Space does not permit us to supply the complete proof, which follows this outline:

1. The modified actor/critic algorithm behaves exactly the same as a Q-learning algorithm constrained by a particular exploration strategy.

2. Q-learning converges to $V^*$ with probability one, given the conditions above (Tsitsiklis, 1993; Jaakkola *et al*, 1993; Watkins & Dayan, 1992).

3. Therefore, the modified actor/critic algorithm also converges to $V^*$ with probability one.

The commutative diagram below illustrates how the modified actor/critic algorithm behaves exactly like Q-learning constrained by a particular exploration strategy. The function $H$ recovers Q-values from the policy $\pi$ and value function $\hat{V}$. $H^{-1}$ recovers $(\pi, \hat{V})$ from the Q-values, thus determining an exploration strategy. Given the ability to move back and forth between $(\pi, \hat{V})$ and $\hat{Q}$, we can determine how to change $(\pi, \hat{V})$ by converting to $\hat{Q}$, determining updated Q-values, and then converting back to obtain an updated $(\pi, \hat{V})$. The modified actor/critic algorithm simply collapses this process into one step, bypassing the explicit use of Q-values.

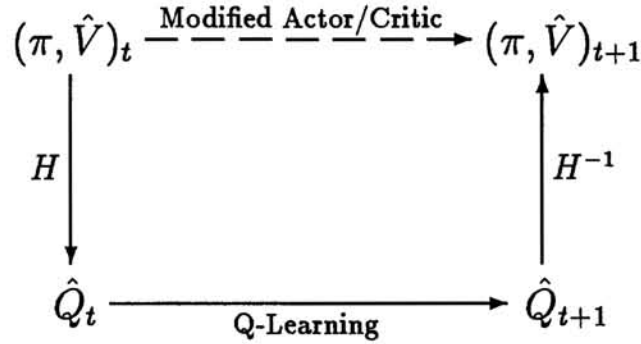

Following the diagram above, $(\pi, \hat{V})$ can be converted to Q-values as follows:

$$\hat{Q}(x, a) = H_{\hat{V}(x)}(z_a).$$

Going the other direction, Q-values can be converted to $(\pi, \hat{V})$ as follows:

$$\hat{V}(x) = max_{a \in A_x} \hat{Q}(x, a) \quad \text{and} \quad p_a = \frac{H^{-1}_{\hat{V}(x)}[\hat{Q}(x,a)]}{\sum_{b \in A_x} H^{-1}_{\hat{V}(x)}[\hat{Q}(x,b)]}.$$

The only Q-value that should change at time $t$ is the one corresponding to the state/action pair that was visited at time $t$; call it $\hat{Q}(x, i)$. In order to prove the convergence theorem, we must verify that after an iteration of the modified actor/critic algorithm, its encoded Q-values match the values produced by Q-learning:

$$\hat{Q}_{t+1}(x, a) = \hat{Q}_t(x, i) + \alpha_{xi}(n) \, [r + \gamma \max_{b \in A_y} \hat{Q}_t(y, b) - \hat{Q}_t(x, i)], \quad a = i. \qquad (2)$$

$$\hat{Q}_{t+1}(x, a) = \hat{Q}_t(x, a), \quad a \neq i. \qquad (3)$$

In verifying this, it is necessary to consider the four cases where $\hat{Q}(x, i)$ is, or is not, the maximum Q-value for state $x$ at times $t$ and $t + 1$. Only enough space exists to present a detailed verification of one case.

**Case 1:** $\hat{Q}_t(x, i) = max \; \hat{Q}_t(x, \cdot) \quad \text{and} \quad \hat{Q}_{t+1}(x, i) = max \; \hat{Q}_{t+1}(x, \cdot).$

In this case, $p_i(t) = p_{max}(t)$ and $p_i(t + 1) = p_{max}(t + 1)$, since $H_{\hat{V}_t(x)}$ and $H_{\hat{V}_{t+1}(x)}$ are strictly increasing. Therefore $z_i(t) = 1$ and $z_i(t + 1) = 1$. Therefore, $\hat{V}_t(x) = H_{\hat{V}_t(x)}[1] = H_{\hat{V}_t(x)}[z_i(t)] = \hat{Q}_t(x, i)$, and

$$
\begin{aligned}
\hat{Q}_{t+1}(x,i) &= H_{\hat{V}_{t+1}(x)}[z_i(t+1)] \\
&= H_{\hat{V}_{t+1}(x)}[1] \\
&= \hat{V}_{t+1}(x) \\
&= \hat{V}_t(x) + \alpha_{xi}(n)\,\varepsilon \\
&= \hat{Q}_t(x,i) + \alpha_{xi}(n)\,[r + \gamma \max_{b \in A_y} \hat{Q}_t(y,b) - \hat{Q}_t(x,i)].
\end{aligned}
$$

This establishes (2). To show that (3) holds, we have that

$$
\begin{aligned}
\hat{V}_{t+1}(x) &= \hat{V}_t(x) + \alpha_{xi}(n)\,\varepsilon \\
&= \hat{Q}_t(x,i) + \alpha_{xi}(n)\,\varepsilon \\
&= H_{\hat{V}_t(x)}[z_i(t)] + \alpha_{xi}(n)\,\varepsilon \\
&= H_{\hat{V}_t(x)}[H^{-1}_{\hat{V}_t(x)}[H_{\hat{V}_t(x)}[z_i(t)] + \alpha_{xi}(n)\,\varepsilon]] \\
&= H_{\hat{V}_t(x)}[z_i^\star(t)]
\end{aligned}
\tag{4}
$$

and

$$
\begin{aligned}
\hat{Q}_{t+1}(x,a) &= H_{\hat{V}_{t+1}(x)}[z_a(t+1)] \\
&= H_{\hat{V}_{t+1}(x)}\Big[\frac{p_a(t+1)}{p_{max}(t+1)}\Big] \\
&= H_{\hat{V}_{t+1}(x)}\Big[\frac{z_a(t)/N}{z_i^\star(t)/N}\Big] \quad if \ a \neq i \\
&= H_{\hat{V}_{t+1}(x)}\Big[\frac{z_a(t)}{z_i^\star(t)}\Big] \\
&= H_{H_{\hat{V}_t(x)}[z_i^\star(t)]}\Big[\frac{z_a(t)}{z_i^\star(t)}\Big] \quad by \ (4) \\
&= H_{\hat{V}_t(x)}[z_a(t)] \quad by \ a \ property \ of \ H \\
&= \hat{Q}_t(x,a).
\end{aligned}
$$

The other cases can be shown similarly.

## 7 CONCLUSIONS

We have presented an actor/critic algorithm that is equivalent to Q-learning constrained by a particular exploration strategy. Like Q-learning, it estimates $V^*$ directly without a model of the underlying decision process. It uses exactly the same amount of storage as Q-learning: one location for every state/action pair. (For each state, $|A| - 1$ locations are needed to store the action probabilities, since they must sum to one. The remaining location can be used to store the value of that state.) One advantage of Q-learning is that its exploration is uncoupled from its value function estimates. In the modified actor/critic algorithm, the exploration strategy is more constrained.

It is still an open question whether other actor/critic algorithms are guaranteed to converge. One way to approach this question would be to investigate further the relationship between the modified actor/critic algorithm described here and the actor/critic algorithms that have been employed by others.

## Acknowledgements

We thank Vijay Gullapalli and Rich Sutton for helpful discussions. This research was supported by Air Force Office of Scientific Research grant F49620-93-1-0269.

## References

A. G. Barto, S. J. Bradtke & S. P. Singh. (1993) Learning to act using real-time dynamic programming. *Artificial Intelligence*, Accepted.

A. G. Barto, R. S. Sutton & C. W. Anderson. (1983) Neuronlike adaptive elements that can solve difficult learning control problems. *IEEE Transactions on Systems, Man, and Cybernetics* **13**:835-846.

A. G. Barto, R. S. Sutton & C. J. C. H. Watkins. (1990) Learning and sequential decision making. In M. Gabriel & J. Moore, editors, *Learning and Computational Neuroscience: Foundations of Adaptive Networks*. MIT Press, Cambridge, MA.

D. P. Bertsekas & J. N. Tsitsiklis. (1989) *Parallel and Distributed Computation: Numerical Methods*. Prentice-Hall, Englewood Cliffs, NJ.

T. Jaakkola, M. I. Jordan & S. P. Singh. (1993) On the convergence of stochastic iterative dynamic programming algorithms. MIT Computational Cognitive Science Technical Report 9307.

L. Lin. (1993) *Reinforcement Learning for Robots Using Neural Networks*. PhD Thesis, Carnegie Mellon University, Pittsburgh, PA.

A. L. Samuel. (1963) Some studies in machine learning using the game of checkers. In E. Feigenbaum & J. Feldman, editors, *Computers and Thought*. McGraw-Hill, New York, NY.

R. S. Sutton. (1988) Learning to predict by the methods of temporal differences. *Machine Learning* **3**:9-44.

R. S. Sutton. (1990) Integrated architectures for learning, planning, and reacting based on approximating dynamic programming. In *Proceedings of the Seventh International Conference on Machine Learning*.

J. N. Tsitsiklis. (1994) Asynchronous stochastic approximation and Q-learning. *Machine Learning* **16**:185-202.

C. J. C. H. Watkins. (1989) *Learning from Delayed Rewards*. PhD thesis, Cambridge University.

C. J. C. H. Watkins & P. Dayan. (1992) Q-learning. *Machine Learning* **8**:279-292.

R. J. Williams & L. C. Baird. (1993) Analysis of some incremental variants of policy iteration: first steps toward understanding actor-critic learning systems. Technical Report NU-CCS-93-11. Northeastern University College of Computer Science.
